# Active Learning for Function Approximation

**Kah Kay Sung**
(sung@ai.mit.edu)
Massachusetts Institute of Technology
Artificial Intelligence Laboratory
545 Technology Square
Cambridge, MA 02139

**Partha Niyogi**
(pn@ai.mit.edu)
Massachusetts Institute of Technology
Artificial Intelligence Laboratory
545 Technology Square
Cambridge, MA 02139

## Abstract

We develop a principled strategy to sample a function optimally for function approximation tasks within a Bayesian framework. Using ideas from optimal experiment design, we introduce an objective function (incorporating both bias and variance) to measure the degree of approximation, and the potential utility of the data points towards optimizing this objective. We show how the general strategy can be used to derive precise algorithms to select data for two cases: learning unit step functions and polynomial functions. In particular, we investigate whether such active algorithms can learn the target with fewer examples. We obtain theoretical and empirical results to suggest that this is the case.

## 1 INTRODUCTION AND MOTIVATION

*Learning from examples* is a common supervised learning paradigm that hypothesizes a target concept given a stream of training examples that describes the concept. In function approximation, *example-based learning* can be formulated as synthesizing an approximation function for data sampled from an unknown target function (Poggio and Girosi, 1990).

*Active learning* describes a class of example-based learning paradigms that seeks out new training examples from specific regions of the input space, instead of passively accepting examples from some data generating source. By judiciously selecting ex-

amples instead of allowing for possible random sampling, *active learning* techniques can conceivably have faster learning rates and better approximation results than passive learning methods.

This paper presents a Bayesian formulation for *active learning* within the function approximation framework. Specifically, here is the problem we want to address: Let $D_n = \{(x_i, y_i)|i = 1, \ldots, n\}$ be a set of $n$ data points sampled from an unknown target function $g$, possibly in the presence of noise. Given an approximation function concept class, $\mathcal{F}$, where each $f \in \mathcal{F}$ has prior probability $\mathcal{P}_{\mathcal{F}}[f]$, one can use regularization techniques to approximate $g$ from $D_n$ (in the Bayes optimal sense) by means of a function $\hat{g} \in \mathcal{F}$. We want a strategy to determine at what input location one should sample the next data point, $(x_{N+1}, y_{N+1})$, in order to obtain the "best" possible Bayes optimal approximation of the unknown target function $g$ with our concept class $\mathcal{F}$.

The data sampling problem consists of two parts:

**1) Defining what we mean by the "best" possible Bayes optimal approximation of an unknown target function.** In this paper, we propose an optimality criterion for evaluating the "goodness" of a solution with respect to an *unknown* target function.

**2) Formalizing precisely the task of determining where in input space to sample the next data point.** We express the above mentioned optimality criterion as a cost function to be minimized, and the task of choosing the next sample as one of minimizing the cost function with respect to the input space location of the next sample point.

Earlier work (Cohn, 1991; MacKay, 1992) have tried to use similar optimal experiment design (Fedorov, 1972) techniques to collect data that would provide maximum information about the target function. Our work differs from theirs in several respects. First, we use a different, and perhaps more general, optimality criterion for evaluating solutions to an unknown target function, based on a measure of function uncertainty that incorporates both bias and variance components of the total *output* generalization error. In contrast, MacKay and Cohn use only variance components in model parameter space. Second, we address the important sample complexity question, i.e., does the active strategy require fewer examples to learn the target to the same degree of uncertainty? Our results are stated in PAC-style (Valiant, 1984). After completion of this work, we learnt that Sollich (1994) had also recently developed a similar formulation to ours. His analysis is conducted in a statistical physics framework.

The rest of the paper is organized as follows: Section 2, develops our active sampling paradigm. In Sections 3 and 4, we consider two classes of functions for which active strategies are obtained, and investigate their performance both theoretically and empirically.

## 2    THE MATHEMATICAL FRAMEWORK

In order to optimally select examples for a learning task, one should first have a clear notion of what an "ideal" learning goal is for the task. We can then measure an example's utility in terms of how well the example helps the learner achieve the

goal, and devise an active sampling strategy that selects examples with maximum potential utility. In this section, we propose one such learning goal — to find an approximation function $\hat{g} \in \mathcal{F}$ that "best" estimates the *unknown* target function $g$. We then derive an example utility cost function for the goal and finally present a general procedure for selecting examples.

## 2.1 EVALUATING A SOLUTION TO AN UNKNOWN TARGET — THE EXPECTED INTEGRATED SQUARED DIFFERENCE

Let $g$ be the target function that we want to estimate by means of an approximation function $\hat{g} \in \mathcal{F}$. If the target function $g$ were known, then one natural measure of how well (or badly) $\hat{g}$ approximates $g$ would be the *Integrated Squared Difference* (ISD) of the two functions:

$$\delta(\hat{g}, g) = \int_{x_{lo}}^{x_{hi}} (g(x) - \hat{g}(x))^2 dx. \tag{1}$$

In most function approximation tasks, the target $g$ is unknown, so we clearly cannot express the quality of a learning result, $\hat{g}$, in terms of $g$. We can, however, obtain an *expected* integrated squared difference (EISD) between the *unknown* target, $g$, and its estimate, $\hat{g}$, by treating the unknown target $g$ as a random variable from the approximation function concept class $\mathcal{F}$. Taking into account the $n$ data points, $D_n$, seen so far, we have the following a-posteriori likelihood for $g$: $P(g|D_n) \propto \mathcal{P}_{\mathcal{F}}[g]P(D_n|g)$. The *expected* integrated squared difference (EISD) between an unknown target, $g$, and its estimate, $\hat{g}$, given $D_n$, is thus:

$$E_{\mathcal{F}}[\delta(\hat{g}, g)|D_n] = \int_{g \in \mathcal{F}} P(g|D_n)\delta(\hat{g}, g)dg = \int_{g \in \mathcal{F}} \mathcal{P}_{\mathcal{F}}[g]P(D_n|g)\delta(\hat{g}, g)dg. \tag{2}$$

## 2.2 SELECTING THE NEXT SAMPLE LOCATION

We can now express our learning goal as minimizing the *expected integrated squared difference* (EISD) between the unknown target $g$ and its estimate $\hat{g}$. A reasonable sampling strategy would be to choose the next example from the input location that minimizes the EISD between $g$ and the new estimate $\hat{g}_{n+1}$. How does one predict the new EISD that results from sampling the next data point at location $x_{n+1}$ ?

Suppose we also know the target output value (possibly noisy), $y_{n+1}$, at $x_{n+1}$. The EISD between $g$ and its new estimate $\hat{g}_{n+1}$ would then be $E_{\mathcal{F}}[\delta(\hat{g}_{n+1}, g)|D_n \cup (x_{n+1}, y_{n+1})]$, where $\hat{g}_{n+1}$ can be recovered from $D_n \cup (x_{n+1}, y_{n+1})$ via regularization. In reality, we do not know $y_{n+1}$, but we can derive for it the following conditional probability distribution:

$$P(y_{n+1}|x_{n+1}, D_n) \propto \int_{f \in \mathcal{F}} P(D_n \cup (x_{n+1}, y_{n+1})|f)\mathcal{P}_{\mathcal{F}}[f]df. \tag{3}$$

This leads to the following *expected* value for the new EISD, if we sample our next data point at $x_{n+1}$:

$$\mathcal{U}(\hat{g}_{n+1}|D_n, x_{n+1}) = \int_{-\infty}^{\infty} P(y_{n+1}|x_{n+1}, D_n)E_{\mathcal{F}}[\delta(\hat{g}_{n+1}, g)|D_n \cup (x_{n+1}, y_{n+1})]dy_{n+1}. \tag{4}$$

Clearly, the optimal input location to sample next is the location that minimizes the cost function in Equation 4 (henceforth referred to as the *total output uncertainty*), i.e.,

$$\hat{x}_{n+1} = \arg\min_{x_{n+1}} \mathcal{U}(g_{n+1}|D_n, x_{n+1}). \tag{5}$$

## 2.3 SUMMARY OF ACTIVE LEARNING PROCEDURE

We summarize the key steps involved in finding the optimal next sample location:

**1)** Compute $P(g|D_n)$. This is the a-posteriori likelihood of the different functions $g$ given $D_n$, the $n$ data points seen so far.

**2)** Fix a new point $x_{n+1}$ to sample.

**3)** Assume a value $y_{n+1}$ for this $x_{n+1}$. One can compute $P(g|D_n \cup (x_{n+1}, y_{n+1}))$ and hence the *expected* integrated squared difference between the target and its new estimate. This is given by $E_{\mathcal{F}}[\delta(\hat{g}_{n+1}, g)|D_n \cup (x_{n+1}, y_{n+1})]$. See also Equation 2.

**4)** At the given $x_{n+1}$, $y_{n+1}$ has a probability distribution given by Equation 3. Averaging over all $y_{n+1}$'s, we obtain the *total output uncertainty* for $x_{n+1}$, given by $\mathcal{U}(\hat{g}_{n+1}|D_n, x_{n+1})$ in Equation 4.

**5)** Sample at the input location that minimizes the *total output uncertainty* cost function.

## 3   EXAMPLE 1: UNIT STEP FUNCTIONS

To demonstrate the usefulness of the above procedure, let us first consider the following simple class of indicator functions parameterized by a single parameter $a$ which takes values in $[0, 1]$. Thus

$$\mathcal{F} = \{1_{[a,1]}|0 \le a \le 1\}$$

We obtain a prior $P(g = 1_{[a,1]})$ by assuming that $a$ has an a-priori uniform distribution on $[0, 1]$. Assume that data, $D_n = \{(x_i; y_i); i = 1, ..n\}$ consistent with some unknown target function $1_{[a_t,1]}$ (which the learner is to approximate) has been obtained. We are interested in choosing a point $x \in [0, 1]$ to sample which will provide us with maximal information. Following the general procedure outlined above we go through the following steps.

For ease of notation, let $x_R$ be the right most point belonging to $D_n$ whose $y$ value is 0, i.e., $x_R = \max_{i=1,..n}\{x_i|y_i = 0\}$. Similarly let $x_L = \min_{i=1,..n}\{x_i|y_i = 1\}$ and let $x_L - x_R = w$.

**1)** We first need to get $P(g|D_n)$. It is easy to show that

$$P(g = 1_{[a,1]}|D_n) = \frac{1}{w} \text{ if } a \in [x_R, x_L]; 0 \text{ otherwise}$$

**2)** Suppose we sample next at a particular $x \in [0, 1]$, we would obtain $y$ with the distribution

$$P(y = 0|D_n, x) = \frac{(x_L - x)}{x_L - x_R} = \frac{(x_L - x)}{w} \text{ if } x \in [x_R, x_L]; 1 \text{ if } x \le x_R; 0 \text{ otherwise}$$

For a particular $y$, the new data set would be $D_{n+1} = D_n \cup (x, y)$ and the corresponding EISD can be easily obtained using the distribution $P(g|D_{n+1})$. Averaging this over $P(y|D_n, x)$ as in step 4 of the general procedure, we obtain

$$\mathcal{U}(\hat{g}_{n+1}|D_n, x) = \begin{cases} w^2/12 & \text{if } x \le x_R \text{ or } x \ge x_L \\ (1/12w)((x_L - x)^3 + (x - x_R)^3) & \text{otherwise} \end{cases}$$

Clearly the point which minimizes the expected *total output uncertainty* is the midpoint of $x_L$ and $x_R$.
$$\hat{x}_{n+1} = \arg \min_{x \in [0,1]} \mathcal{U}(g|D_n, x) = (x_L + x_R)/2$$

Thus applying the general procedure to this special case reduces to a binary search learning algorithm which queries the midpoint of $x_R$ and $x_L$. An interesting question at this stage is whether such a strategy provably reduces the sample complexity; and if so, by how much. It is possible to prove the following theorem which shows that for a certain pre-decided *total output uncertainty* value, the active learning algorithm takes fewer examples to learn the target to the same degree of *total output uncertainty* than a random drawing of examples according to a uniform distribution.

**Theorem 1** *Suppose we want to collect examples so that we are guaranteed with high probability (i.e. probability $> 1 - \delta$) that the* total output uncertainty *is less than $\epsilon$. Then a passive learner would require at least $\frac{1}{\sqrt{(48\epsilon)}} \ln(1/\delta)$ examples while the active strategy described earlier would require at most $(1/2) \ln(1/12\epsilon)$ examples.*

## 4 · EXAMPLE 2: THE CASE OF POLYNOMIALS

In this section we turn our attention to a class of univariate polynomials (from $[-5, 5]$ to $\Re$) of maximum degree $K$, i.e.,
$$\mathcal{F} = \{g(a_0, \ldots, a_K) = \sum_{i=0}^{K} a_i x^i\}$$

As before, the prior on $\mathcal{F}$ is obtained here by assuming a prior on the parameters; in particular we assume that $\mathbf{a} = (a_0, a_1, \ldots, a_K)$ has a multivariate normal distribution $N(0, \mathcal{S})$. For simplicity, it is assumed that the parameters are independent, i.e., $\mathcal{S}$ is a diagonal matrix with $\mathcal{S}_{i,i} = \sigma_i^2$. In this example we also incorporate noise (distributed normally according to $N(0, \sigma^2)$). As before, there is a target $g_t \in G$ which the learner is to approximate on the basis of data. Suppose the learner is in possession of a data set $D_n = \{(x_i, y_i = g_t(x_i) + \eta); i = 1 \ldots n\}$ and is to receive another data point. The two options are 1) to sample the function at a point $x$ according to a uniform distribution on the domain $[-5, 5]$ (passive learning) and 2) follow our principled active learning strategy to select the next point to be sampled.

### 4.1 ACTIVE STRATEGY

Here we derive an exact expression for $\hat{x}_{n+1}$ (the next query point) by applying the general procedure described earlier. Going through the steps as before,

**1)** It is possible to show that $P(g(\mathbf{a})|D_n) = P(\mathbf{a}|D_n)$ is again a multivariate normal distribution $N(\mu, \Sigma_n)$ where $\mu = \sum_{i=1}^{N} y_i \mathbf{x_i}$, $\mathbf{x_i} = (1, x_i, x_i^2, \ldots, x_i^K)^T$ and
$$\Sigma_n^{-1} = \mathcal{S}^{-1} + \frac{1}{2\sigma^2} \sum_{i=1}^{n} (\mathbf{x_i} \mathbf{x_i}^T)$$

**2)** Computation of the *total output uncertainty* $\mathcal{U}(\hat{g}_{n+1}|D_n, x)$ requires several steps. Taking advantage of the Gaussian distribution on both the parameters $\mathbf{a}$ and the noise, we obtain (see Niyogi and Sung, 1995 for details):
$$\mathcal{U}(g|D_n, x) = |\Sigma_{n+1} \mathbf{A}|$$

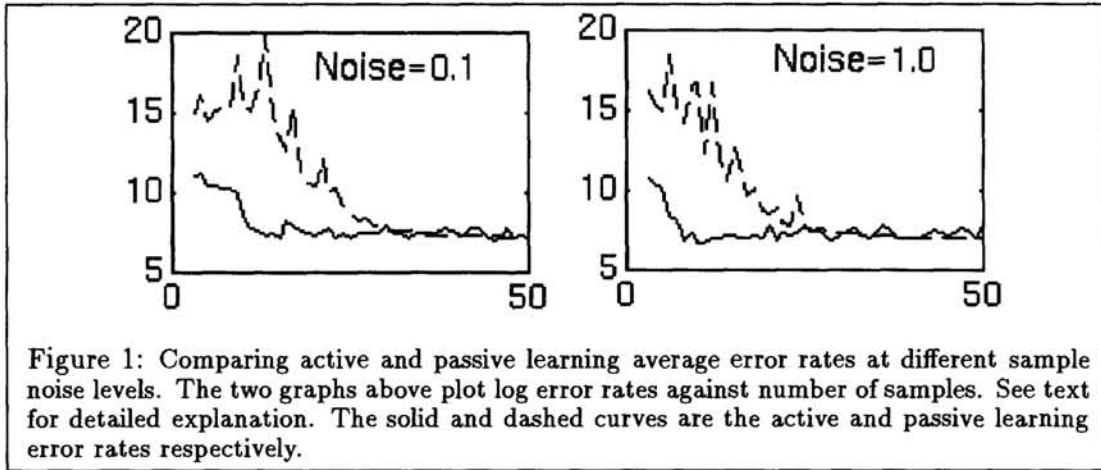

Figure 1: Comparing active and passive learning average error rates at different sample noise levels. The two graphs above plot log error rates against number of samples. See text for detailed explanation. The solid and dashed curves are the active and passive learning error rates respectively.

where $\mathbf{A}$ is a matrix of numbers whose $i, j$ element is $\int_{-5}^{5} t^{(i+j-2)} dt$. $\Sigma_{n+1}$ has the same form as $\Sigma_n$ and depends on the previous data, the priors, noise and $x_{n+1}$. When minimized over $x_{n+1}$, we get $\hat{x}_{n+1}$ as the maximum utility location where the active learner should next sample the unknown target function.

## 4.2 SIMULATIONS

We have performed several simulations to compare the performance of the active strategy developed in the previous section to that of a passive learner (who receives examples according to a uniform random distribution on the domain $[-5, 5]$). The following issues have been investigated.

**1) Average error rate as a function of the number of examples:** Is it indeed the case that the active strategy has superior error performance for the same number of examples? To investigate this we generated 1000 test target polynomial functions (of maximum degree 9) according to the following Gaussian prior on the parameters: for each $a_i$, $P(a_i) = N(0, 0.9^i)$. For each target polynomial, we collected data according to the active strategy as well as the passive (random) strategy for varying number of data points. Figure 1 shows the average error rate (i.e., the integrated squared difference between the actual target function and its estimate, averaged over the 1000 different target polynomials) as a function of the number of data points. Notice that the active strategy has a lower error rate than the passive for the same number of examples and is particularly true for small number of data. The active strategy uses the same priors that generate the test target functions. We show results of the same simulation performed at two noise levels (noise standard deviation 0.1 and 1.0). In both cases the active strategy outperforms the passive learner indicating robustness in the face of noise.

**2) Incorrect priors:** How sensitive is the active learner to possible differences between its prior assumptions on the class $\mathcal{F}$ and the true priors? We repeated the function learning task of the earlier case with the test targets generated in the same way as before. The active learner assumes a slightly different Gaussian prior and polynomial degree from the target ($\text{Std}(a_i) = 0.7^i$ and $K = 7$ for the active learner versus $\text{Std}(a_i) = 0.8^i$ and $K = 8$ for the target). Despite its inaccurate priors, the

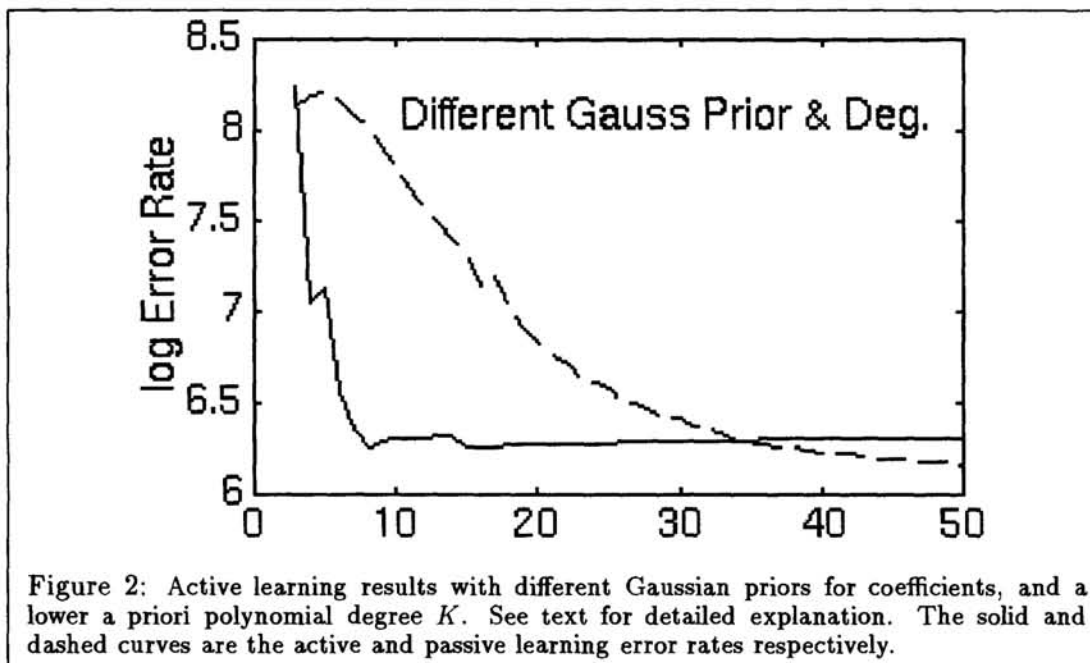

Figure 2: Active learning results with different Gaussian priors for coefficients, and a lower a priori polynomial degree $K$. See text for detailed explanation. The solid and dashed curves are the active and passive learning error rates respectively.

active learner outperforms the passive case.

**3) The distribution of points:** How does the active learner choose to sample the domain for maximally reducing uncertainty? There are a few sampling trends which are noteworthy here. First, the learner does not simply sample the domain on a uniform grid. Instead it chooses to cluster its samples typically around $K + 1$ locations for concept classes with maximum degree $K$ as borne out by simulations where $K$ varies from 5 to 9. One possible explanation for this is it takes only $K + 1$ points to determine the target in the absence of noise. Second, as the noise increases, although the number of clusters remains fixed, they tend to be distributed away from the origin. It seems that for higher noise levels, there is less pressure to fit the data closely; consequently the prior assumption of lower order polynomials dominates. For such lower order polynomials, it is profitable to sample away from the origin as it reduces the variance of the resulting fit. (Note the case of linear regression).

**Remarks**
1) Notice that because the class of polynomials is linear in its model parameters, **a**, the new sample location $(\hat{x}_{n+1})$ does not depend on the $y$ values actually observed but only on the $x$ values sampled. Thus if the learner is to collect $n$ data points, it can pre-compute the $n$ points at which to sample from the start. In this sense the active algorithm is not really adaptive. This behavior has also been observed by MacKay (1992) and Sollich (1994).
2) Needless to say, the general framework from optimal design can be used for any function class within a Bayesian framework. We are currently investigating the possibility of developing active strategies for Radial Basis Function networks. While it is possible to compute exact expressions for $\hat{x}_{n+1}$ for such RBF networks with fixed centers, for the case of moving centers, one has to resort to numerical minimization. For lack of space we do not include those results in this paper.

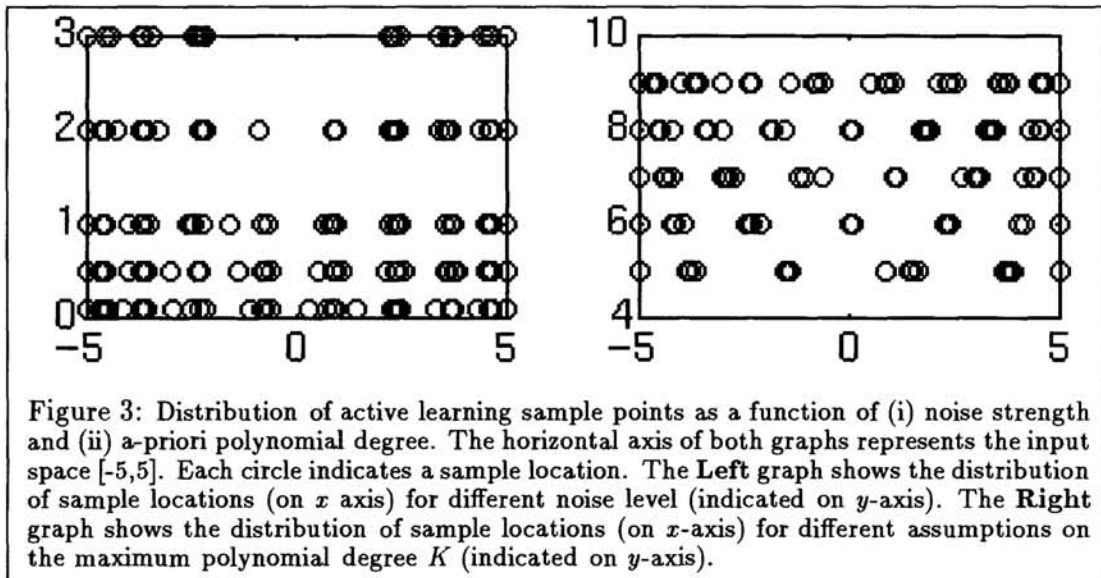

Figure 3: Distribution of active learning sample points as a function of (i) noise strength and (ii) a-priori polynomial degree. The horizontal axis of both graphs represents the input space [-5,5]. Each circle indicates a sample location. The **Left** graph shows the distribution of sample locations (on $x$ axis) for different noise level (indicated on $y$-axis). The **Right** graph shows the distribution of sample locations (on $x$-axis) for different assumptions on the maximum polynomial degree $K$ (indicated on $y$-axis).

## 5   CONCLUSIONS

We have developed a Bayesian framework for active learning using ideas from optimal experiment design. Our focus has been to investigate the possibility of improved sample complexity using such active learning schemes. For a simple case of unit step functions, we are able to derive a binary search algorithm from a completely different standpoint. Such an algorithm then provably requires fewer examples for the same error rate. We then show how to derive specific algorithms for the case of polynomials and carry out extensive simulations to compare their performance against the benchmark of a passive learner with encouraging results. This is an application of the optimal design paradigm to function learning and seems to bear promise for the design of more efficient learning algorithms.

**References**

D. Cohn. (1991) A Local Approach to Optimal Queries. In D. Touretzky (ed.), *Proc. of 1990 Connectionist Summer School*, San Mateo, CA, 1991. Morgan Kaufmann Publishers.

V. Fedorov. (1972) *Theory of Optimal Experiments*. Academic Press, New York, 1972.

D. MacKay. (1992) *Bayesian Methods for Adaptive Models*. PhD thesis, CalTech, 1992.

P. Niyogi and K. Sung. (1995) Active Learning for Function Approximation: Paradigms from Optimal Experiment Design. Tech Report AIM–1483, AI Lab., MIT, In Preparation.

M. Plutowski and H. White. (1991) Active Selection of Training Examples for Network Learning in Noiseless Environments. Tech Report CS91-180, Dept. of Computer Science and Engineering, University of California, San Diego, 1991.

T. Poggio and F. Girosi. (1990) Regularization Algorithms for Learning that are Equivalent to Multilayer Networks. *Science*, 247:978–982, 1990.

P. Sollich. (1994) Query Construction, Entropy, Generalization in Neural Network Models. *Physical Review E*, 49:4637–4651, 1994.

L. Valiant. (1984) A Theory of Learnable. *Proc. of the 1984 STOC*, p436–445, 1984.
